# Multimodular Architecture for Remote Sensing Operations.

Sylvie Thiria[1,2]

Carlos Mejia[1]

Fouad Badran[1,2]

Michel Crépon[3]

[1] Laboratoire de Recherche en Informatique
Université de Paris Sud, B 490 - 91405 ORSAY Cedex France

[2] CEDRIC, Conservatoire National des Arts et Métiers
292 rue Saint Martin - 75 003 PARIS

[3] Laboratoire d'Océanographie et de Climatologie (LODYC)
T14 Université de PARIS 6 - 75005 PARIS (FRANCE)

## Abstract

This paper deals with an application of Neural Networks to satellite remote sensing observations. Because of the complexity of the application and the large amount of data, the problem cannot be solved by using a single method. The solution we propose is to build multi-modules NN architectures where several NN cooperate together. Such system suffer from generic problem for whom we propose solutions. They allow to reach accurate performances for multi-valued function approximations and probability estimations. The results are compared with six other methods which have been used for this problem. We show that the methodology we have developed is general and can be used for a large variety of applications.

## 1   INTRODUCTION

Neural Networks have been used for many years to solve hard real world applications which involve large amounts of data. Most of the time, these problems cannot be solved with a unique technique and involve successive processing of the input data. Sophisticated NN architectures have thus been designed to provide good performances e.g. [Lecun et al. 90]. However this approach is limited for many reasons : the design of these architectures requires a lot of a priori knowledge about the task and is complicated. Such NN are difficult to train because of their large size and are dedicated to a specific problem. Moreover if the task is slightly modified, these NN have to be entirely redesigned and retrained. It is our feeling that complex problems cannot be solved efficiently with a single NN whatever sophisticated it is. A more fruitful approach is to use modular architectures where several simple NN modules cooperate together. This methodology is far more general and allows to easily build very sophisticated architectures which are able to handle the different processing steps which are necessary for example in speech or signal processing. These architectures can be easily modified to incorporate some additional knowledge about the problem or some changes in its specifications.

We have used these ideas to build a multi-module NN for a satellite remote sensing application. This is a hard problem which cannot be solved by a single NN. The different modules of our architecture are thus dedicated to specific tasks and allow to perform successive processing of the data. This approach allows to take into account in successive steps different informations about the problem. Furthermore, errors which may occur at the output of some modules may be corrected by others which allows to reach very good performances. Making these different modules cooperate raises several problems which appear to be generic for these architectures. It is thus interesting to study different solutions for their design, training, and the efficient information exchanges between modules. In the present paper, we first briefly describe the geophysical problem and its difficulties, we then present the different modules of our architecture and their cooperation, we compare our results to those of several other methods and discuss the advantages of our method.

## 2   THE GEOPHYSICAL PROBLEM

Scatterometers are active microwave radars which accurately measure the power of transmitted and backscatter signal radiations in order to compute the normalized radar cross section ($\sigma_0$) of the ocean surface. The $\sigma_0$ depends on the wind speed, the incidence angle $\theta$ (which is the angle between the radar beam and the vertical at the illuminated cell) and the azimuth angle (which is the horizontal angle $\chi$ between the wind and the antenna of the radar). The empirically based relationship between $\sigma_0$ and the local wind vector can be established which leads to the determination of a geophysical model function.

The model developed by A. Long gives a more precise form to this functional. It has been shown that for an angle of incidence $\theta$, the general expression for $\sigma_0$ can be satisfactorily represented by a Fourrier series :

$$\sigma_0 = U.\left(\frac{1 + b_1.\cos\chi + b_2.\cos 2\chi}{1 + b_1 + b_2}\right),$$

(1)

with $U = A.v^\gamma$

Long's model specifies that A and $\gamma$ only depend on the angle of incidence $\theta$, and that $b_1$ and $b_2$ are a function of both the wind speed v and the angle of incidence $\theta$ (Figure 1).

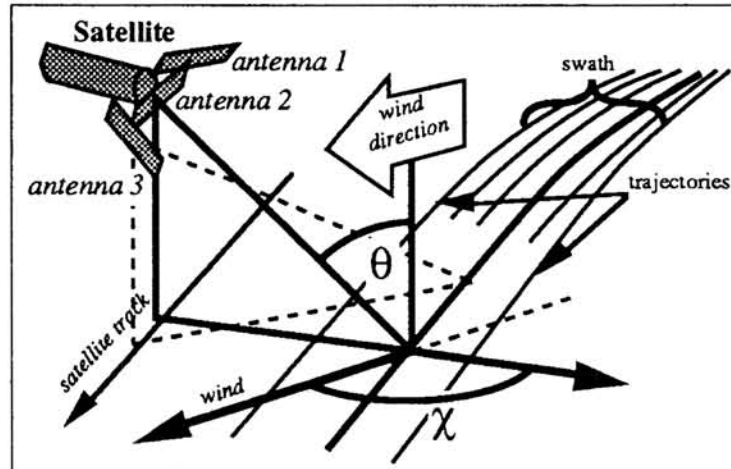

Figure 1 : Definition of the different geophysical scales.

*For now, the different parameters b1, b2 A and $\gamma$ used in this model are determined experimentally.*

Conversely it becomes possible to compute the wind direction by using several antenna with different orientations with respect to the satellite track. The geophysical model function (1) can then be inverted using the three measurements of $\sigma_0$ given by the three antennas, it computes wind vector (direction and speed). Evidence shows that for a given trajectory within the swath (Figure 1) i.e. $(\theta_1, \theta_2, \theta_3)$ fixed, $\theta_i$ being the incidence angle of the beam linked to antenna *i*, the functional F is of the form presented in Fig.2 .

In the absence of noise, the determination of the wind direction would be unique in most cases. Noise-free ambiguities arise due to the bi-harmonic nature of the model function with respect to $\chi$. The functional F presents singular points. At constant wind speed F yields a Lissajous curve ; in the singular points the direction is ambiguous with respect to the triplet measurements $(\sigma_1, \sigma_2, \sigma_3)$ as it is seen in Fig. 2. At these points F yields two directions differing by 160°. In practice, since the backscatter signal is noisy the number and the frequency of ambiguities is increased.

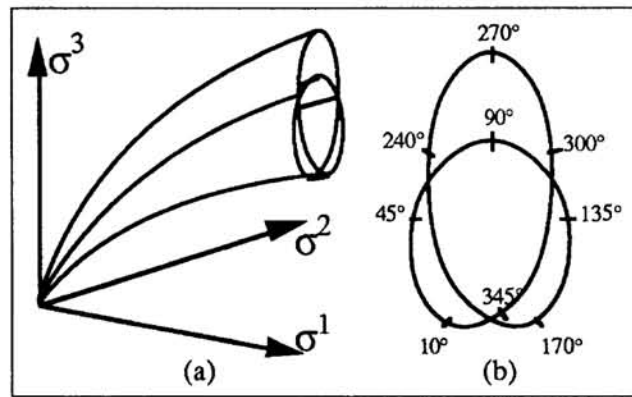

Figure 2 : (a) Representation of the Functional **F** for a given trajectory (b) Graphics
obtained for a section of (a) at constant wind speed.

*The problem is therefore how to set up an accurate (exact) wind map using the observed
measurements $(\sigma_1, \sigma_2, \sigma_3)$.*

## 3   THE METHOD

We propose to use multi-layered quasi-linear networks (MLP) to carry out this inversion
phase.  Indeed these nets are able of approximate complex non-linear functional relations;
it becomes possible by using a set of measurements to determine **F** and to realize the
inversion.

The determination of the wind's speed and direction lead to two problems of different
complexity,  each of them is solved using a dedicated multi-modular system.  The two
modules are then linked together to build a two level architecture.  To take into account
the strong dependence of the measurements with respect to the trajectory, each module (or
level ) consists of $n$ distinct but similar systems, a specific system being dedicated to each
satellite trajectory ($n$ being the number of trajectories in a swath (Figure 1)).

The first level will allow the determination of the wind speed at every point of the swath.
The results obtained will then be supplied to the second level as supplementary data
which allow to compute the wind direction.  Thus, we propose a two-level architecture
which constitutes an automatic method for the computation of wind maps (Figure 3).
The computation is performed sequentially between the different levels, each one
supplying the next with the parameters needed.

Owing to the space variability of the wind, the measurements at a point are closely related
to those performed  in the neighbourhood.  Taking into account this context must
therefore bring important supplementary information to dealiase the ambiguities. At a
point, the input data for a given system are therefore the measurements observed at that
point and at it's eight closest neighbours.

All the networks used by the different systems are MLP trained with the back-propagation
algorithm.  The successive modifications were performed using a second order stochastic
gradient : which is the approximation of the Levenberg-Marquardt rule.

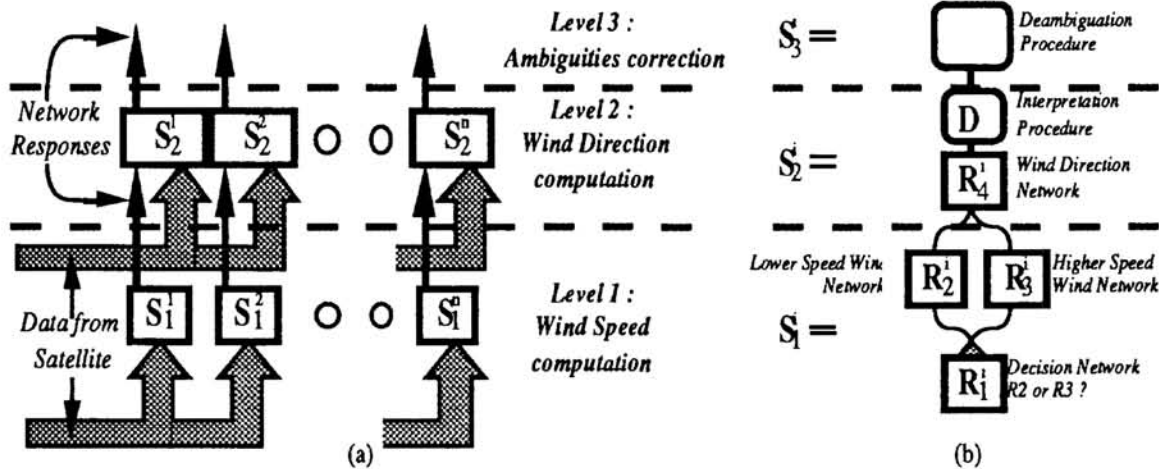

Figure 3 : The three systems S1, S2 and S3 for a given trajectory.

One system is dedicated to a proper trajectory. As a result the networks used on the same level of the global architecture are of the same type; only the learning set numerical values change from one system to another. Each network learning set will therefore consist of the data mesured on its trajectory. We present here the results for the central trajectory, performances for the others are similar.

## 3.1   THE NETWORK DECODING : FIRST LEVEL

A system (S1) in the first level allows to compute the wind speed (in $ms^{-1}$) along a trajectory. Because the function $F_1$ to be learned (signal $\rightarrow$ wind speed) is highly non-linear, each system is made of three networks (see Figure 3) : R1 allows to decide the range of the wind speed ($4 \leq v < 12$ or $12 \leq v < 20$); according to the R1 output an accurate value is computed using R2 for the first range and R3 for the other. The first level is built from 10 of these systems (one for each trajectory).

Each network (R1, R2, R3) consists of four fully connected layers. For a given point, we have introduced the knowledge of the radar measurements at the neighbouring points. The same experiments were performed without introducing this notion of vicinity, the learning and test performances were reduced by 17%, which proves the advantages of this approach. The input layer of each network consists of 27 automata : these 9x3 automata correspond to the $\sigma_0$ values relative to each antenna for the point to be considered and its eight neighbours.

R1 output layer has two cells : one for $4 \leq v < 12$ and the other for $12 \leq v < 20$; so its 4 layers are respectively built of 27, 25, 25, 2 automata.

R2 and R3 compute the exact wind speed. The output layer is represented by a unique output automaton and codes this wind speed v at the point considered between $[-1, +1]$. The four layers of each network are respectively formed of 27, 25, 25, 1 automata.

## 3.2   DECODING THE DIRECTION : SECOND LEVEL

Now the function $F_2$ (signal $\rightarrow$ wind direction) has to be learned. This level is located after the first one, so the wind speed has already been computed at all points. For each trajectory a system S2 allows to compute the wind direction, it is made of an MLP and a Decision Direction Process (we call it D). As for F1 we used for each point a contextual information. Thus, the input layer of the MLP consists of 30 automata : the first 9x3 correspond to the $\sigma_0$ values for each antenna, the last three represent three times the first level computed wind speed. However, because the original function has major ambiguities it is more convenient to compute, for a given input, several output values with their probabilities. For this reason we have discretized the desired output. It has been coded in degrees and 36 possible classes have been considered, each representing a 10° interval (between 0° and 360°). So, the MLP is four layered with respectively 30, 25, 25, 36 automata. It can be shown, according to the coding of the desired output, that the network approximates Bayes discriminant function or Bayes probability distribution related to the discretized transfer function $F_2$ [White, 89]. The interpretation of the MLP outputs using the D process allows to compute with accuracy the required function $F_2$. The network outputs represents the 36 classes corresponding to the 36 10° intervals. For a given input, a computed output is a $\mathcal{R}^{36}$ vector whose components can be interpreted to predict the wind direction in degrees. Each component, which is a Bayes discriminant function approximation, can be used as a coefficient of likelihood for each class. The Decision Direction Process D (see Fig. 3) computes real directions using this information. It performs the interpolation of the peaks' curve. D gives for each peak ist wind direction with its coefficients of likelihood.

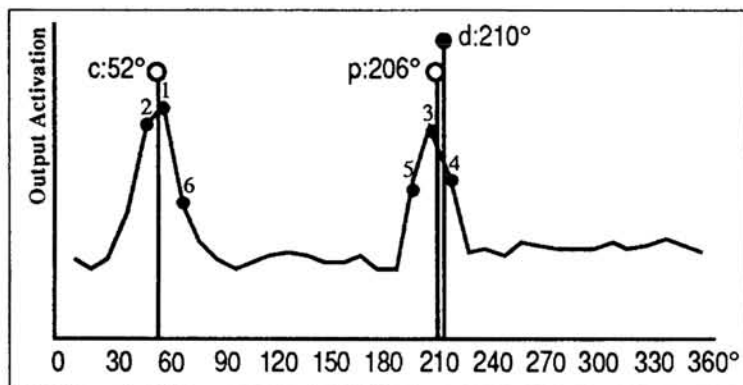

Figure 4 : network's output. The points in the x-axis correspond to the 36 outputs. Each represents an interval of 10° between 0 and 360°. The Y-axis points give the automata computed output. The point indicated by a d corresponds to the desired output angle, c is the most likely solution proposed by D and p is the second one.

The computed wind speed and the most likely wind direction computed by the first two levels allow to build a complete map which still includes errors in the directions. As we have seen in section 2, the physical problem has intrinsic ambiguities, they appear in the results (table 2). The removal of these errors is done by a third level of NN.

## 3.3    CORRECTING THE REMAINING ERRORS : THIRD LEVEL

This problem has been dealt with in [Badran & al 91] and is not discussed here. The method is related to image processing using MLP as optimal filter. The use of different filters taking into account the 5x5 vicinities of the point considered permits to detect the erroneous directions and to choose among the alternative proposed solutions. This method enables to correct up to 99.5% of the errors.

# 4    RESULTS

As actual data does not exist yet, we have tested the method on values computed from real meteorological models. The swaths of the scatterometer ERS1 were simulated by flying a satellite on wind fields given by the ECMWF forecasting model. The sea roughness values $(\sigma_1,\sigma_2,\sigma_3)$ given by the three antennas were computed by inverting the Long model. Noise was then added to the simulated measurements in order to reproduce the errors made by the scatterometer. (A gaussian noise of zero average and of standard deviation 9.5% for both lateral antennas and 8.7% for the central antenna was added at each measurement).Twenty two maps obtained for the southern Atlantic Ocean were used to establish the learning sets. The 22 maps were selected randomly during the 30 days of September 1985 and nine remaining maps were used for the tests.

## 4.1    DECODING THE SPEED : FIRST LEVEL

In the results presented in Table 1, a predicted measurement is considered correct if it differs from the desired output by 1 m/s. It has to be noticed that the oceanographer's specification is 2 m/s; the present results illustrate the precision of the method.

Table 1 : performances on the wind speed

| Performances | | performances | bias |
|---|---|---|---|
| Accuracy 1 m/s | learning | 99.3% | 0.045m/s |
| | test | 98.4 % | 0.038m/s |

## 4.2    DECODING THE DIRECTION : SECOND LEVEL

It is found that good performances are obtained after the interpretation of the best two peaks only. When it is compared to usual methods which propose up to six possible directions, this method appears to be very powerful. Table 2 shows the performances using one or two peaks. The function **F** and its singularities have been recovered with a good accuracy, the noise added during the simulations in order to reproduce the noise made by the measuring devices has been removed.

Table 2 : performances on the wind direction using the complete system

| Performances | | one peak | two peaks |
|---|---|---|---|
| Precision 20° | learning | 68.0 % | 99.1 % |
| | test | 72.0 % | 99.2 % |

## 5   VALIDATION OF THE RESULTS

In order to prove the power of the NN approach, table 3 compare our results with six classical methods [Chi & Li 88].

Table 3 shows that the NN results are very good compared to other techniques, moreover all the classical methods are based on the assumption that a precise analytical function $((v,\chi) \rightarrow \sigma)$ exists, the NN method is more general and does not depend on such an assumption. Moreover the decoding of a point with NN requires approximately 23 ms on a SUN4 working station. This time is to be compared with the 0.25 second necessary for the decoding by present methods.

Table 3 : performances simulation results $E_{rms}$ (in m/s) for different fixed wind speed

| Speed | WLSL | ML | LS | WLS | AWLS | L1 | LWSS | N.N |
|-------|------|------|------|------|------|------|------|------|
| Low | 0.92 | 0.66 | 0.67 | 0.74 | 0.69 | 0.63 | 1.02 | 0.49 |
| Middle | 0.89 | 0.85 | 1.10 | 1.31 | 0.89 | 0.98 | 0.87 | 0.53 |
| Hight | 3.71 | 3.44 | 4.11 | 5.52 | 3.52 | 4.06 | 3.49 | 1.18 |

The wind vector error e is defined as follows : $e = V1 - V2$ where V1 is the true wind vector and V2 is the estimated wind vector, $E_{rms} = E(\| e \|)$.

## 6   CONCLUSION

Performances reached when processing satellite remote sensing observations have proved that multi-modular architectures where simple NN modules cooperate can cope with real world applications. The methodology we have developed is general and can be used for a large variety of applications, it provides solutions to generic problems arising when dealing with NN cooperation.

### References

Badran F, Thiria S, Crepon M (1991) : Wind ambiguity removal by the use of neural network techniques, *J.G.R Journal of Geophysical Research vol 96 n°C 11 p 20521-20529, November 15.*

Chong-Yung C, Fuk K Li (1969) : A Comparative Study of Several Wind Estimation Algorithms for Spacebornes scatterometers. *IEEE transactions on geoscience and remote sensing, vol 26, No 2.*

Le Cun Y., Boser B., & al., (1990) : Handwritten Digit Recognition with a Back-Propagation Network- in D.Touretzky (ed.) *Advances in Neural Information Processing Systems 2* , 396-404, Morgan Kaufmann

White H. (1989) : Learning in Artificial Neural Networks : A Statistical Perspective. *Neural Computation, 1, 425-464.*
